# Efficient Optimization for Discriminative Latent Class Models

**Armand Joulin**[*]
INRIA
23, avenue d'Italie,
75214 Paris, France.
armand.joulin@inria.fr

**Francis Bach**[*]
INRIA
23, avenue d'Italie,
75214 Paris, France.
francis.bach@inria.fr

**Jean Ponce**[*]
Ecole Normale Supérieure
45, rue d'Ulm
75005 Paris, France.
jean.ponce@ens.fr

## Abstract

Dimensionality reduction is commonly used in the setting of multi-label supervised classification to control the learning capacity and to provide a meaningful representation of the data. We introduce a simple forward probabilistic model which is a multinomial extension of reduced rank regression, and show that this model provides a probabilistic interpretation of discriminative clustering methods with added benefits in terms of number of hyperparameters and optimization. While the expectation-maximization (EM) algorithm is commonly used to learn these probabilistic models, it usually leads to local maxima because it relies on a non-convex cost function. To avoid this problem, we introduce a local approximation of this cost function, which in turn leads to a quadratic non-convex optimization problem over a product of simplices. In order to maximize quadratic functions, we propose an efficient algorithm based on convex relaxations and low-rank representations of the data, capable of handling large-scale problems. Experiments on text document classification show that the new model outperforms other supervised dimensionality reduction methods, while simulations on unsupervised clustering show that our probabilistic formulation has better properties than existing discriminative clustering methods.

## 1 Introduction

Latent representations of data are wide-spread tools in supervised and unsupervised learning. They are used to reduce the dimensionality of the data for two main reasons: on the one hand, they provide numerically efficient representations of the data; on the other hand, they may lead to better predictive performance. In supervised learning, latent models are often used in a generative way, e.g., through mixture models on the input variables only, which may not lead to increased predictive performance. This has led to numerous works on supervised dimension reduction (e.g., [1, 2]), where the final discriminative goal of prediction is taken explicitly into account during the learning process.

In this context, various probabilistic models have been proposed, such as mixtures of experts [3] or discriminative restricted Boltzmann machines [4], where a layer of hidden variables is used between the inputs and the outputs of the supervised learning model. Parameters are usually estimated by expectation-maximization (EM), a method that is computationally efficient but whose cost function may have many local maxima in high dimensions.

In this paper, we consider a simple *discriminative latent class* (DLC) model where inputs and outputs are independent given the latent representation.We make the following contributions:

---

[*]WILLOW project-team, Laboratoire d'Informatique de l'Ecole Normale Supérieure, (ENS/INRIA/CNRS UMR 8548).

– We provide in Section 2 a quadratic (non convex) local approximation of the log-likelihood of our model based on the EM auxiliary function. This approximation is optimized to obtain robust initializations for the EM procedure.
– We propose in Section 3.3 a novel probabilistic interpretation of discriminative clustering with added benefits, such as fewer hyperparameters than previous approaches [5, 6, 7].
– We design in Section 4 a low-rank optimization method for non-convex quadratic problems over a product of simplices. This method relies on a convex relaxation over completely positive matrices.
– We perform experiments on text documents in Section 5, where we show that our inference technique outperforms existing supervised dimension reduction and clustering methods.

## 2    Probabilistic discriminative latent class models

We consider a set of $N$ observations $x_n \in \mathbb{R}^p$, and their labels $y_n \in \{1, \ldots, M\}$, $n \in \{1, \ldots, N\}$. We assume that each observation $x_n$ has a certain probability to be in one of $K$ latent classes, modeled by introducing hidden variables $z_n \in \{1, \ldots, K\}$, and that these classes should be predictive of the label $y_n$. We model directly the conditional probability of $z_n$ given the input data $x_n$ and the probability of the label $y_n$ given $z_n$, while making the assumption that $y_n$ and $x_n$ are independent given $z_n$ (leading to the directed graphical model $x_n \rightarrow z_n \rightarrow y_n$). More precisely, we assume that, given $x_n$, $z_n$ follows a multinomial logit model while, given $z_n$, $y_n$ is a multinomial variable:

$$p(z_n = k | x_n) = \frac{e^{w_k^T x_n + b_k}}{\sum_{j=1}^{K} e^{w_j^T x_n + b_j}} \quad \text{and} \quad p(y_n = m | z_n = k) = \alpha_{km}, \qquad (1)$$

with $w_k \in \mathbb{R}^p$, $b_k \in \mathbb{R}$ and $\sum_{m=1}^{M} \alpha_{km} = 1$. We use the notation $w = (w_1, \ldots, w_K)$, $b = (b_1, \ldots, b_K)$ and $\alpha = (\alpha_{km})_{1 \leq k \leq K, 1 \leq m \leq M}$. Note that the model defined by (1) can be kernelized by replacing implicitly or explicitly $x$ by the image $\Phi(x)$ of a non linear mapping.

**Related models.**    The simple two-layer probabilistic model defined in Eq. (1), can be interpreted and compared to other methods in various ways. First, it is an instance of a mixture of experts [3] where each expert has a constant prediction. It has thus weaker predictive power than general mixtures of experts; however, it allows efficient optimization as shown in Section 4. It would be interesting to extend our optimization techniques to the case of experts with non-constant predictions. This is what is done in [8] where a convex relaxation of EM for a similar mixture of experts is considered. However, [8] considers the maximization with respect to hidden variables rather than their marginalization, which is essential in our setting to have a well-defined probabilistic model. Note also that in [8], the authors derive a convex relaxation of the softmax regression problems, while we derive a quadratic approximation. It is worth trying to combine the two approaches in future work.

Another related model is a two-layer neural network. Indeed, if we marginalize the latent variable $z$, we get that the probability of $y$ given $x$ is a linear combination of softmax functions of linear functions of the input variables $x$. Thus, the only difference with a two-layer neural network with softmax functions for the last layer is the fact that our last layer considers linear parameterization in the mean parameters rather than in the natural parameters of the multinomial variable. This change allows us to provide a convexification of two-layer neural networks in Section 4.

Among probabilistic models, a discriminative restricted Boltzmann machine (RBM) [4, 9] models $p(y|z)$ as a softmax function of linear functions of $z$. Our model assumes instead that $p(y|z)$ is linear in $z$. Again, this distinction between mean parameters and natural parameters allows us to derive a quadratic approximation of our cost function. It would of course be of interest to extend our optimization technique to the discriminative RBM.

Finally, one may also see our model as a multinomial extension of reduced-rank regression (see, e.g. [10]), which is commonly used with Gaussian distributions and reduces to singular value decomposition in the maximum likelihood framework.

## 3 Inference

We consider the negative conditional log-likelihood of $y_n$ given $x_n$ (regularized in $w$ to avoid overfitting) where $\theta = (\alpha, w, b)$ and $y_{nm}$ is equal to 1 if $y_n = m$ and 0 otherwise:

$$\ell(\theta) = -\frac{1}{N}\sum_{n=1}^{N}\sum_{m=1}^{M} y_{nm}\log p(y_{nm} = 1|x_n) + \frac{\lambda}{2K}\|w\|_F^2. \tag{2}$$

### 3.1 Expectation-maximization

A popular tool for solving maximum likelihood problems is the EM algorithm [10]. A traditional way of viewing EM is to add auxiliary variables and minimize the following upperbound of the negative log-likelihood $\ell$, obtained by using the concavity of the logarithm:

$$F(\xi, \theta) = -\frac{1}{N}\sum_{n=1}^{N}\sum_{m=1}^{M} y_{nm}\left[\sum_{k=1}^{K} \xi_{nk}\log\frac{y_n^T\alpha_k e^{w_k^T x_n + b_k}}{\xi_{nk}} - \log\left(\sum_{k=1}^{K} e^{w_k^T x_n + b_k}\right)\right] + \frac{\lambda}{2K}\|w\|_F^2,$$

where $\alpha_k = (\alpha_{k1}, \ldots, \alpha_{km})^T \in \mathbb{R}^M$ and $\xi = (\xi_1, \ldots, \xi_K)^T \in \mathbb{R}^{N \times K}$ with $\xi_n = (\xi_{n1}, \ldots, \xi_{nK}) \in \mathbb{R}^K$. The EM algorithm can be viewed as a two-step block-coordinate descent procedure [11], where the first step (E-step) consists in finding the optimal auxiliary variables $\xi$, given the parameters of the model $\theta$. In our case, the result of this step is obtained in closed form as $\xi_{nk} \propto y_n^T\alpha_k e^{w_k^T x_n + b_k}$ with $\xi_n^T 1_K = 1$. The second step (M-step) consists of finding the best set of parameters $\theta$, given the auxiliary variables $\xi$. Optimizing the parameters $\alpha_k$ leads to the closed form updates $\alpha_k \propto \sum_{n=1}^{N}\xi_{nk}y_n$ with $\alpha_k^T 1_M = 1$ while optimizing jointly on $w$ and $b$ leads to a softmax regression problem, which we solved with Newton method.

Since $F(\xi, \theta)$ is not jointly convex in $\xi$ and $\theta$, this procedure stops when it reaches a local minimum, and its performance strongly depends on its initialization. We propose in the following section, a robust initialization for EM given our latent model, based on an approximation of the auxiliary cost function obtained with the M-step.

### 3.2 Initialization of EM

Minimizing $F$ w.r.t. $\xi$ leads to the original log-likelihood $\ell(\theta)$ depending on $\theta$ alone. Minimizing $F$ w.r.t. $\theta$ gives a function of $\xi$ alone. In this section, we focus on deriving a quadratic approximation of this function, which will be minimized to obtain an initialization for EM.

We consider second-order Taylor expansions around the value of $\xi$ corresponding to the uniformly distributed latent variables $z_n$, independent of the observations $x_n$, i.e., $\xi_0 = \frac{1}{K}1_N 1_K^T$. This choice is motivated by the lack of a priori information on the latent classes. We briefly explain the calculation of the expansion of the terms depending on $(w, b)$. For the rest of the calculation, see the supplementary material.

**Second-order Taylor expansion of the terms depending on** $(w, b)$**.** Assuming uniformly distributed variables $z_n$ and independence between $z_n$ and $x_n$ implies that $w_k^T x_n + b_k = 0$. Therefore, using the second-order expansion of the log-sum-exp function $\varphi(u) = \log(\sum_{k=1}^{K}\exp(u_k))$ around 0 leads to the following approximation of the terms depending on $(w, b)$:

$$J_{wb}(\xi) = \text{cst} + \frac{K}{2N}\text{tr}(\xi\xi^T) - \frac{1}{2K}\min_{w,b}\left[\frac{1}{N}\|(K\xi - Xw - b)\Pi_K\|_F^2 + \lambda\|w\|_F^2 + O(\|Xw + b\|^3)\right],$$

where $\Pi_K = I - \frac{1}{K}1_K 1_K^T$ is the usual centering projection matrix, and $X = (x_1, \ldots, x_N)^T$. The third-order term $O(\|Xw + b\|_F^3)$ can be replaced by third-order terms in $\|\xi - \xi_0\|$, which makes the minimization with respect to $w$ and $b$ correspond to a multi-label classification problem with a square-loss [7, 10, 12]. Its solution may be obtained in closed form and leads to:

$$J_{wb}(\xi) = \text{cst} + \frac{K}{2N}\text{tr}\left[\xi\xi^T\left(I - A(X, \lambda)\right)\right] + O(\|\xi - \xi_0\|^3),$$

where $A(X, \lambda) = \Pi_N\left(I - X(N\lambda I + X^T\Pi_N)^{-1}X^T\right)\Pi_N$.

**Quadratic approximation.** Omitting the terms that are independent of $\xi$ or of an order in $\xi$ higher than two, the second-order approximation $J_{\text{app}}$ of the function obtained for the M-step is:

$$J_{\text{app}}(\xi) = \frac{K}{2}\text{tr}\Big[\xi\xi^T\Big(B(Y) - A(X,\lambda)\Big)\Big], \tag{3}$$

where $B(Y) = \frac{1}{N}\Big(Y(Y^TY)^{-1}Y^T - \frac{1}{N}1_N1_N^T\Big)$ and $Y \in \mathbb{R}^{N \times M}$ is the matrix with entries $y_{nm}$.

**Link with ridge regression.** The first term, $\text{tr}(\xi\xi^T B(Y))$, is a concave function in $\xi$, whose maximum is obtained for $\xi\xi^T = I$ (each variable in a different cluster). The second term, $A(X,\lambda)$, is the matrix obtained in ridge regression [7, 10, 12]. Since $A(x,\lambda)$ is a positive semi-definite matrix such that $A(X,\lambda)1_N = 0$, the maximum of the second term is obtained for $\xi\xi^T = 1_N1_N^T$ (all variables in the same cluster). $J_{\text{app}}(\xi)$ is thus a combination of a term trying to put every point in the same cluster and a term trying to spread them equally. Note that in general, $J_{\text{app}}$ is not convex.

**Non linear predictions.** Using the matrix inversion lemma, $A(X,\lambda)$ can be expressed in terms of the Gram matrix $K = XX^T$, which allows us to use any positive definite kernel in our framework [12], and tackle problems that are not linearly separable. Moreover, the square loss gives a natural interpretation of the regularization parameter $\lambda$ in terms of the implicit number of parameters of the learning procedure [10]. Indeed, the *degree of freedom* defined as $df = n(1 - \text{tr}A)$ provides a intuitive method for setting the value of $\lambda$ [7, 10].

**Initialization of EM.** We optimize $J_{\text{app}}(\xi)$ to get a robust initialization for EM. Since the entries of each vector $\xi_n$ sum to 1, we optimize $J_{\text{app}}$ over a set of $N$ simplices in $K$ dimensions, $\mathcal{S} = \{v \in \mathbb{R}^K \mid v \geq 0,\ v^T1_K = 1\}$. However, since this function is not convex, minimizing it directly leads to local minima. We propose, in Section 4, a general reformulation of any non-convex quadratic program over a set of $N$ simplices and propose an efficient algorithm to optimize it.

### 3.3 Discriminative clustering

The goal of clustering is to find a low-dimensional representation of unlabeled observations, by assigning them to $K$ different classes, Xu et al. [5] proposes a discriminative clustering framework based on the SVM and [7] simplifies it by replacing the hinge loss function by the square loss, leading to ridge regression. By taking $M = N$ and the labels $Y = I$, we obtain a formulation similar to [7] where we are looking for a latent representation that can recover the identity matrix. However, unlike [5, 7], our discriminative clustering framework is based on a probabilistic model which may allow natural extensions. Moreover, our formulation naturally avoids putting all variables in the same cluster, whereas [5, 7] need to introduce constraints on the size of each cluster. Also, our model leads to a soft assignment of the variables, allowing flexibility in the shape of the clusters, whereas [5, 7] is based on hard assignment. Finally, since our formulation is derived from EM, we obtain a natural rounding by applying the EM algorithm after the optimization whereas [7] uses a coarse k-means rounding. Comparisons with these algorithms can be found in Section 5.

## 4 Optimization of quadratic functions over simplices

To initialize the EM algorithm, we must minimize the *non-convex* quadratic cost function defined by Eq. (3) over a product of $N$ simplices. More precisely, we are interested in the following problems:

$$\min_V\ f(V) = \tfrac{1}{2}\text{tr}\left(VV^T B\right) \quad \text{s.t.} \quad V = (V_1,\ldots,V_N)^T \in \mathbb{R}^{N \times K} \quad \text{and} \quad \forall n,\ V_n \in \mathcal{S}, \tag{4}$$

where $B$ can be any $N \times N$ symmetric matrix. Denoting $v = \text{vec}(V) \in \mathbb{R}^{NK}$ the vector obtained by stacking all the columns of $V$ and defining $Q = (B^T \otimes I_K)^T$, where $\otimes$ is the Kronecker product [13], the problem (4) is equivalent to:

$$\min_v\ \tfrac{1}{2}\, v^T Q v \quad \text{s.t.} \quad v \in \mathbb{R}^{NK},\ v \geq 0 \quad \text{and} \quad (I_N \otimes 1_K^T)v = 1_N. \tag{5}$$

Note that this formulation is general, and that $Q$ could be any $NK \times NK$ symmetric matrix. Traditional convex relaxation methods [14] would rewrite the objective function as $v^T Q v = \text{tr}(Qvv^T) =$

$\text{tr}(QT)$ where $T = vv^T$ is a rank-one matrix which satisfies the set of constraints:

$$- \quad T \in \mathcal{DN}_K = \{T \in \mathbb{R}^{NK \times NK} \mid T \geq 0,\ T \succcurlyeq 0\} \tag{6}$$

$$- \quad \forall\, n, m \in \{1, \ldots, N\},\ 1_K^T T_{nm} 1_K = 1, \tag{7}$$

$$- \quad \forall\, n, i, j \in \{1, \ldots, N\},\ T_{ni} 1_K = T_{nj} 1_K. \tag{8}$$

We note $\mathcal{F}$ the set of matrix $T$ verifying (7-8). With the unit-rank constraint, optimizing over $v$ is exactly equivalent to optimizing over $T$. The problem is relaxed into a convex problem by removing the rank constraint, leading to a semidefinite programming problem (SDP) [15].

**Relaxation.** Optimizing $T$ instead of $v$ is computationally inefficient since the running time complexity of general purpose SDP toolboxes is in this case $O\big((KN)^7\big)$. On the other hand, for problems without pointwise positivity, [16, 17] have considered low-rank representations of matrices $T$, of the form $T = VV^T$ where $V$ has more than one column. In particular, [17] shows that the non convex optimization with respect to $V$ leads to the global optimum of the convex problem in $T$.

In order to apply the same technique here, we need to deal with the pointwise nonnegativity. This can be done by considering the set of *completely positive matrices*, i.e.,

$$\mathcal{CP}_K = \{T \in \mathbb{R}^{NK \times NK} \mid \exists R \in \mathbb{N}^*,\ \exists V \in \mathbb{R}^{NK \times R},\ V \geq 0,\ T = VV^T\}.$$

This set is *strictly* included in the set $\mathcal{DN}_K$ of *doubly non-negative matrices* (i.e., both pointwise nonnegative and positive semi-definite). For $R \geq 5$, it turns out that the intersection of $\mathcal{CP}_K$ and $\mathcal{F}$ is the convex hull of the matrices $vv^T$ such that $v$ is an element of the product of simplices [16]. This implies that the convex optimization problem of minimizing $\text{tr}\,(QT)$ over $\mathcal{CP}_K \cap \mathcal{F}$ is equivalent to our original problem (for which no polynomial-time algorithm is known).

However, even if the set $\mathcal{CP}_K \cap \mathcal{F}$ is convex, optimizing over it is computationally inefficient [18]. We thus follow [17] and consider the problem through the low-rank pointwise nonnegative matrix $V \in \mathbb{R}^{NK \times R}$ instead of through matrices $T = VV^T$. Note that following arguments from [16], if $R$ is large enough, there are no local minima. However, because of the positivity constraint one cannot find in polynomial time a local minimum of a differentiable function. Nevertheless, any gradient descent algorithm will converge to a stationary point. In Section 5, we compare results with $R > 1$ than with $R = 1$, which corresponds to a gradient descent directly on the simplex.

**Problem reformulation.** In order to derive a local descent algorithm, we reformulate the constraints (7-8) in terms of $V$ (details can be found in the supplementary material). Denoting by $V_r$ the $r$-th column of $V$, $V_r^n$ the $K$-vector such as $V_r = (V_r^1, \ldots, V_r^N)^T$ and $V^n = (V_1^n, \ldots, V_R^n)$, condition (8) is equivalent to $\|V_r^m\|_1 = \|V_r^n\|_1$ for all $n$ and $m$. Substituting this in (7) yields that for all $n$, $\|V^n\|_{2-1} = 1$, where $\|V^n\|_{2-1}^2 = \sum_{r=1}^R (1^T V_r^n)^2$ is the squared $\ell_{2-1}$ norm. We drop this condition by using a rescaled cost function which equivalent. Finally, using the notation $\mathcal{D}$:

$$\mathcal{D} = \{W \in \mathbb{R}^{NK} \mid W \geq 0,\ \forall n, m,\ \|W^n\|_1 = \|W^m\|_1\},$$

we obtain a new equivalent formulation:

$$\min_{V \in \mathbb{R}^{NK \times R},\ \forall r,\ V_r \in \mathcal{D}} \tfrac{1}{2}\text{tr}(VD^{-1}V^T Q) \quad \text{with } D = \text{Diag}((I_N \otimes 1_K)^T VV^T (I_N \otimes 1_K)), \tag{9}$$

where $\text{Diag}(A)$ is the matrix with the diagonal of $A$ and $0$ elsewhere. Since the set of constraints for $V$ is convex, we can use a projected gradient method [19] with the projection step we now describe.

**Projection on $\mathcal{D}$.** Given $N$ $K$-vectors $Z^n$ stacked in a $NK$ vector $Z = [Z^1; \ldots; Z^N]$, we consider the projection of $Z$ on $\mathcal{D}$. For a given positive real number $a$, the projection of $Z$ on the set of all $U \in \mathcal{D}$ such that for all $n$, $\|U^n\|_1 = a$, is equivalent to $N$ independent projections on the $\ell_1$ ball with radius $a$. Thus projecting $Z$ on $\mathcal{D}$ is equivalent to find the solution of:

$$\min_{a \geq 0} \quad L(a) = \sum_{n=1}^N \max_{\lambda_n \in \mathbb{R}} \min_{U^n \geq 0} \tfrac{1}{2}\|U^n - Z^n\|_2^2 + \lambda_n(1_K^T U^n - a),$$

where $(\lambda_n)_{n \leq N}$ are Lagrange multipliers. The problem of projecting each $Z^n$ on the $\ell_1$-ball of radius $a$ is well studied [20], with known expressions for the optimal Lagrange multipliers, $(\lambda_n(a))_{n \leq N}$ and the corresponding projection for a given $a$. The function $L(a)$ is

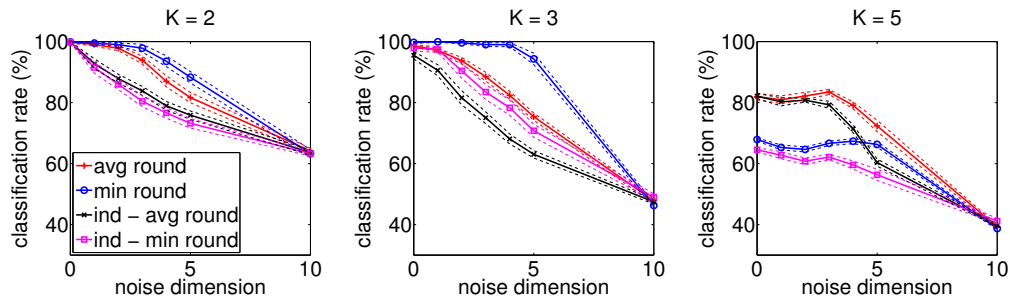

Figure 1: Comparison between our algorithm and $R$ independent optimizations. Also comparison between two rounding: by summing and by taking the best column. Average results for $K = 2, 3, 5$ (Best seen in color).

convex, piecewise-quadratic and differentiable, which yields the first-order optimality condition $\sum_{n=1}^{N} \lambda_n(a) = 0$ for $a$. Several algorithms can be used to find the optimal value of $a$. We use a binary search by looking at the sign of $\sum_{n=1}^{N} \lambda_n(a)$ on the interval $[0, \lambda_{max}]$, where $\lambda_{max}$ is found iteratively. This method was found to be empirically faster than gradient descent.

**Overall complexity and running time.** We use projected gradient descent, the bottleneck of our algorithm is the projection with a complexity of $O(RN^2 K \log(K))$. We present experiments on running times in the supplementary material.

## 5 Implementation and results

We first compare our algorithm with others to optimize the problem (4). We show that the performances are equivalent but, our algorithm can scale up to larger database. We also consider the problem of supervised and unsupervised discriminative clustering. In both cases, we show that our algorithm outperforms existing methods.

**Implementation.** For supervised and unsupervised multilabel classification, we first optimize the second-order approximation $J_{\text{app}}$, using the reformulation (9). We use a projected gradient descent method with Armijo's rule along the projection arc for backtracking [19]. It is stopped after a maximum number of iterations (500) or if relative updates are too small ($10^{-8}$). When the algorithm stops, the matrix $V$ has rank greater than 1 and we use the heuristic $v^* = \sum_{r=1}^{R} V_r \in \mathcal{S}$ as our final solution ("avg round"). We also compare this rounding with another heuristic obtained by taking $v^* = \text{argmin}_{V_r} f(V_r)$ ("min round"). $v^*$ is then used to initialize the EM algorithm described in Section 2.

**Optimization over simplices.** We compare our optimization of the *non-convex* quadratic problem (9) in $V$, to the *convex* SDP in $T = VV^T$ on the set of constraints defined by $T \in \mathcal{DN}_K$, (7) and (8). To optimize the SDP, we use generic algorithms, CVX [21] and PPXA [22]. CVX uses interior points methods whereas PPXA uses proximal methods [22]. Both algorithms are computationally inefficient and do not scale well with either the number of points or the number of constraints. Thus we set $N = 10$ and $K = 2$ on discriminative clustering problems (which are described later in this section). We compare the performances of these algorithms *after* rounding. For the SDP, we take $\xi^* = T1_{NK}$ and for our algorithm we report performances obtained for both rounding discuss above ("avg round" and "min round"). On these small examples, our algorithm associated with "min round" reaches similar performances than the SDP, whereas, associated with "avg round", its performance drops.

**Study of rounding procedures.** We compare the performances of the two different roundings, "min round" and "avg round" on discriminative clustering problems. After rounding, we apply the EM algorithm and look at the classification scores. We also compare our algorithm for a given $R$, to two baselines where we solve independently problem (4) $R$ times and then apply the same roundings ("ind - min round" and "ind - avg round"). Results are shown Figure 1. We consider three

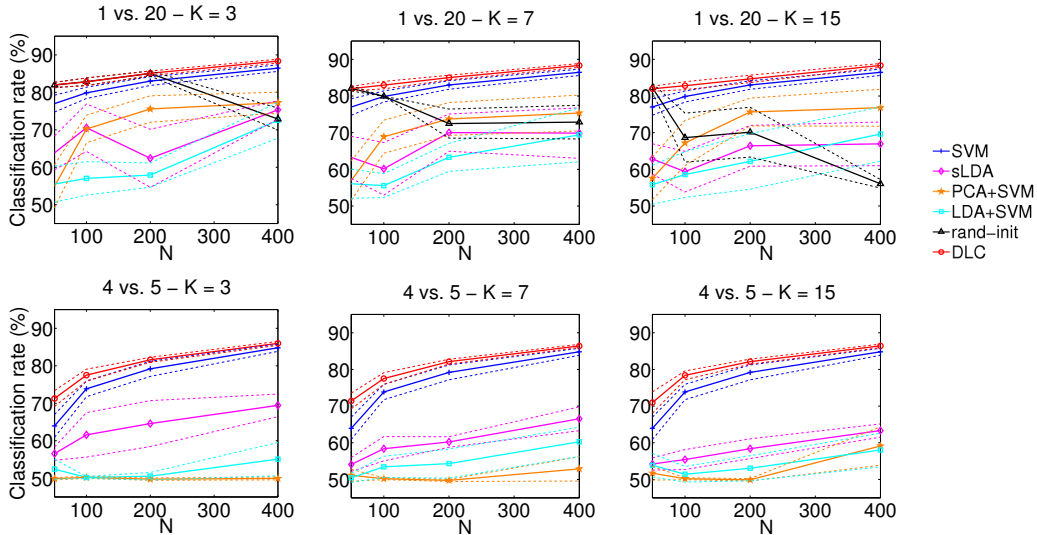

Figure 2: Classification rate for several binary classification tasks (from top to bottom) and for different values of $K$, from left to right (Best seen in color).

different problems, $N = 100$ and $K = 2$, $K = 3$ and $K = 5$. We look at the average performances as the number of noise dimensions increases in discriminative clustering problems. Our method outperforms the baseline whatever rounding we use. Figure 1 shows that on problems with a small number of latent classes ($K < 5$), we obtain better performances by taking the column associated with the lowest value of the cost function ("min round"), than summing all the columns ("avg round"). On the other hand, when dealing with a larger number of classes ($K \geq 5$), the performance of "min round' drops significantly while "avg round" maintains good results. A potential explanation is that summing the columns of $V$ gives a solution close to $\frac{1}{K}1_N 1_K^T$ in expectation, thus in the region where our quadratic approximation is valid. Moreover, the best column of $V$ is usually a local minimum of the quadratic approximation, which we have found to be close to similar local minima of our original problem, therefore, preventing the EM algorithm from converging to another solution. In all others experiments, we choose "avg round".

**Application to classification.** We evaluate the optimization performance of our algorithm (DLC) on text classification tasks. For our experiments, we use the *20 Newsgroups* dataset (http://people.csail.mit.edu/jrennie/), which contains postings to Usenet newsgroups. The postings are organized by content into 20 categories. We use the five binary classification tasks considered in [23, Chapter 4, page 91]. To set the regularization parameter $\lambda$, we use the degree of freedom $df$ (see Section 3.2). Each document has 13312 entries and we take $df = 1000$. We use 50 random initializations for our algorithm. We compare our method with classifiers such as the linear SVM and the supervised Latent Dirichlet Allocation (sLDA) classifier of Blei et al. [2]. We also compare our results to those obtained by an SVM using the features obtained with dimension-reducing methods such as LDA [1] and PCA. For these models, we select parameters with 5-fold cross-validation. We also compare to the EM without our initialization ("rand-init") but also with 50 random initializations, a local descent method which is close to back-propagation in a two-layer neural network, which in this case strongly suffers from local minima problems. An interesting result on computational time is that EM without our initialization needs more steps to obtain a local minimum. It is therefore *slower* than with our initialization in this particular set of experiments. We show some results in Figure 2 (others maybe found in the supplementary material) for different values of $K$ and with an increasing number $N$ of training samples. In the case of topic models, $K$ represents the number of topics. Our method significantly outperforms all the other classifiers. The comparison with "rand-init" shows the importance of our convex initialization. We also note that our performance increases slowly with $K$. Indeed, the number of latent classes needed to correctly separate two classes of text is small. Moreover, the algorithm tends to automatically select $K$. Empirically, we notice that starting with $K = 15$ classes, our average final number of active classes is around 3. This explains the relatively small gain in performance as $K$ increases.

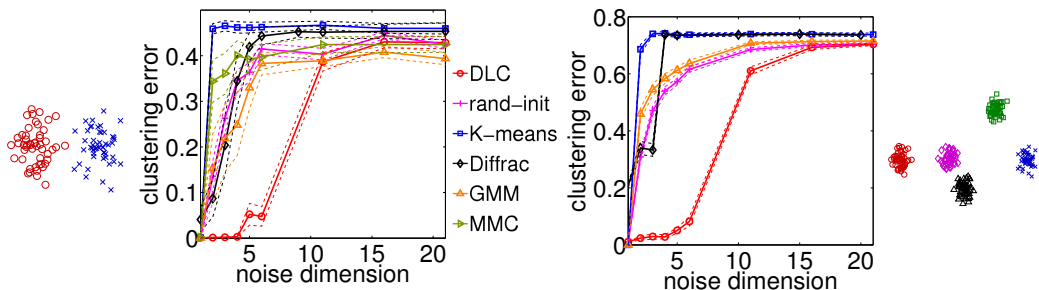

Figure 3: Clustering error when increasing the number of noise dimensions. We have take 50 different problems and 10 random initializations for each of them. $K = 2$, $N = 100$ and $R = 5$, on the left, and $K = 5$, $N = 250$ and $R = 10$, on the right(Best seen in color).

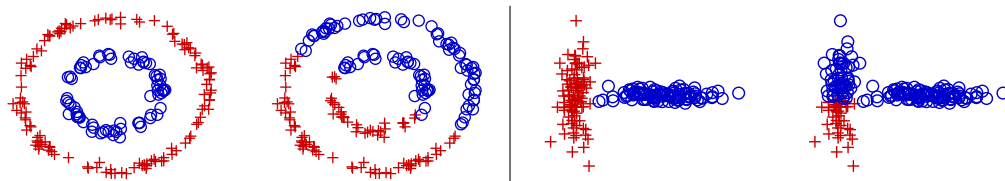

Figure 4: Comparison between our method (left) and k-means (right). First, circles with RBF kernels. Second, linearly separable bumps. $K = 2$, $N = 200$ and $R = 5$ in both cases.

**Application to discriminative clustering.** Figure 3 shows the optimization performance of the EM algorithm with 10 random starting points with ("DLC") and without ("rand-init") our initialization method. We compare their performances to K-means, Gaussian Mixture Model ("GMM"), Diffrac [7] and max-margin clustering ("MMC") [24]. Following [7], we take linearly separable bumps in a two-dimensional space and add dimensions containing random independent Gaussian noise (e.g. "noise dimensions") to the data. We evaluate the ratio of misclassified observations over the total number of observations. For the first experiment, we fix $K = 2$, $N = 100$, and $R = 5$, and for the second $K = 5$, $N = 250$, and $R = 10$. The additional independent noise dimensions are normally distributed. We use linear kernels for all the methods. We set the regularization parameters $\lambda$ to $10^{-2}$ for all experiments but we have seen that results do not change much as long as $\lambda$ is not too small ($> 10^{-8}$). Note that we do not show results for the MMC algorithm when $K = 5$ since this algorithm is specially designed for problems with $K = 2$. It would be interesting to compare to the extension for multi-class problems proposed by Zhang et al. [24]. On both examples, we are significantly better than Diffrac, k-means and MMC. We show in Figure 4 additional examples which are non linearly separable.

# 6 Conclusion

We have presented a probabilistic model for supervised dimension reduction, together with associated optimization tools to improve upon EM. Application to text classification has shown that our model outperforms related ones and we have extended it to unsupervised situations, thus drawing new links between probabilistic models and discriminative clustering. The techniques presented in this paper could be extended in different directions: First, in terms of optimization, while the embedding of the problem to higher dimensions has empirically led to finding better local minima, sharp statements might be made to characterize the robustness of our approach. In terms of probabilistic models, such techniques should generalize to other latent variable models. Finally, some additional structure could be added to the problem to take into account more specific problems, such as multiple instance learning [25], multi-label learning or discriminative clustering for computer vision [26, 27].

**Acknowledgments.** This paper was partially supported by the Agence Nationale de la Recherche (MGA Project) and the European Research Council (SIERRA Project). We would like to thank Toby Dylan Hocking, for his help on the comparison with other methods for the classification task.

# References

[1] David M. Blei, Andrew Y. Ng, Michael I. Jordan, and John Lafferty. Latent Dirichlet Allocation. *Journal of Machine Learning Research*, 3, 2003.

[2] David M. Blei and Jon D. Mcauliffe. Supervised topic models. In *Advances in Neural Information Processing Systems (NIPS)*, 2007.

[3] R. A. Jacobs, M. I. Jordan, S. J. Nowlan, and G. E. Hinton. Adaptive mixtures of local experts. *Neural Computation*, 3(1):79–87, 1991.

[4] H. Larochelle and Y. Bengio. Classification using discriminative restricted boltzmann machines. In *Proceedings of the international conference on Machine learning (ICML)*, 2008.

[5] Linli Xu, James Neufeld, Bryce Larson, and Dale Schuurmans. Maximum margin clustering. In *Advances in Neural Information Processing Systems (NIPS)*, 2004.

[6] Linli Xu. Unsupervised and semi-supervised multi-class support vector machines. In *AAAI*, 2005.

[7] F. Bach and Z. Harchaoui. Diffrac : a discriminative and flexible framework for clustering. In *Advances in Neural Information Processing Systems (NIPS)*, 2007.

[8] N. Quadrianto, T. Caetano, J. Lim, and D. Schuurmans. Convex relaxation of mixture regression with efficient algorithms. In *Advances in Neural Information Processing Systems (NIPS)*, 2009.

[9] G. E. Hinton and R. R. Salakhutdinov. Reducing the dimensionality of data with neural networks. *Science*, 313(5786):504, 2006.

[10] T. Hastie, R. Tibshirani, and J. Friedman. *The Elements of Statistical Learning*. Springer-Verlag, 2001.

[11] David R Hunter and Kenneth Lange. A tutorial on MM algorithms. *The American Statistician*, 58(1):30–37, February 2004.

[12] J Shawe-Taylor and N Cristianini. *Kernel Methods for Pattern Analysis*. Cambridge Univ Press, 2004.

[13] Gene H. Golub and Charles F. Van Loan. *Matrix computations*. Johns Hopkins University Press, 3rd edition, October 1996.

[14] Kurt Anstreicher and Samuel Burer. D.C. versus copositive bounds for standard QP. *Journal of Global Optimization*, 33(2):299–312, October 2005.

[15] S. Boyd and L. Vandenberghe. *Convex Optimization*. Cambridge Univ. Press, 2004.

[16] Samuel Burer. Optimizing a polyhedral-semidefinite relaxation of completely positive programs. *Mathematical Programming Computation*, 2(1):1–19, March 2010.

[17] M. Journée, F. Bach, P.-A. Absil, and R. Sepulchre. Low-rank optimization for semidefinite convex problems. volume 20, pages 2327–2351. SIAM Journal on Optimization, 2010.

[18] A. Berman and N. Shaked-Monderer. *Completely Positive Matrices*. World Scientific Publishing Company, 2003.

[19] D. Bertsekas. *Nonlinear programming*. Athena Scientific, 1995.

[20] P. Brucker. An O(n) algorithm for quadratic knapsack problems. In *Journal of Optimization Theory and Applications*, volume 134, pages 549–554, 1984.

[21] M. Grant and S. Boyd. CVX: Matlab software for disciplined convex programming, version 1.21. *http://cvxr.com/cvx*, August 2010.

[22] Patrick L. Combettes. Solving monotone inclusions via compositions of nonexpansive averaged operators. *Optimization*, 53:475–504, 2004.

[23] Simon Lacoste-Julien. *Discriminative Machine Learning with Structure*. PhD thesis, University of California, Berkeley, 2009.

[24] Kai Zhang, Ivor W. Tsang, and James T. Kwok. Maximum margin clustering made practical. In *Proceedings of the international conference on Machine learning (ICML)*, 2007.

[25] Thomas G. Dietterich and Richard H. Lathrop. Solving the multiple-instance problem with axis-parallel rectangles. *Artificial Intelligence*, 89:31–71, 1997.

[26] P. Felzenszwalb, D. Mcallester, and D. Ramanan. A discriminatively trained, multiscale, deformable part model. In *Proceedings of the Conference on Computer Vision and Pattern Recognition (CVPR)*, 2008.

[27] A. Joulin, F. Bach, and J. Ponce. Discriminative clustering for image co-segmentation. In *Proceedings of the Conference on Computer Vision and Pattern Recognition (CVPR)*, 2010.

